# Approximate Planning in POMDPs with Macro-Actions

**Georgios Theocharous**
MIT AI Lab
200 Technology Square
Cambridge, MA 02139
theochar@ai.mit.edu

**Leslie Pack Kaelbling**
MIT AI Lab
200 Technology Square
Cambridge, MA 02139
lpk@ai.mit.edu

## Abstract

Recent research has demonstrated that useful POMDP solutions do not require consideration of the entire belief space. We extend this idea with the notion of temporal abstraction. We present and explore a new reinforcement learning algorithm over grid-points in belief space, which uses macro-actions and Monte Carlo updates of the Q-values. We apply the algorithm to a large scale robot navigation task and demonstrate that with temporal abstraction we can consider an even smaller part of the belief space, we can learn POMDP policies faster, and we can do information gathering more efficiently.

## 1 Introduction

A popular approach to artificial intelligence is to model an agent and its interaction with its environment through actions, perceptions, and rewards [10]. Intelligent agents should choose actions after every perception, such that their long-term reward is maximized. A well defined framework for this interaction is the partially observable Markov decision process (POMDP) model. Unfortunately solving POMDPs is an intractable problem mainly due to the fact that exact solutions rely on computing a policy over the entire belief-space [6, 3], which is a simplex of dimension equal to the number of states in the underlying Markov decision process (MDP). Recently researchers have proposed algorithms that take advantage of the fact that for most POMDP problems, a large proportion of the belief space is not experienced [7, 9].

In this paper we explore the same idea, but in combination with the notion of temporally extended actions (macro-actions). We propose and investigate a new model-based reinforcement learning algorithm over grid-points in belief space, which uses macro-actions and Monte Carlo updates of the Q-values. We apply our algorithm to large scale robot navigation and demonstrate the various advantages of macro-actions in POMDPs. Our experimental results show that with macro-actions an agent experiences a significantly smaller part of the belief space than with simple primitive actions. In addition, learning is faster because an agent can look further into the future and propagate values of belief points faster. And finally, well designed macros, such as macros that can easily take an agent from a high entropy belief state to a low entropy belief state (e.g., go down the corridor), enable agents to perform information gathering.

## 2  POMDP Planning with Macros

We now describe our algorithm for finding an approximately optimal plan for a known POMDP with macro actions. It works by using a dynamically-created finite-grid approximation to the belief space, and then using model-based reinforcement learning to compute a value function at the grid points. Our algorithm takes as input a POMDP model, a resolution $r$, and a set of macro-actions (described as policies or finite state automata). The output is a set of grid-points (in belief space) and their associated action-values, which via interpolation specify an action-value function over the entire belief space, and therefore a complete policy for the POMDP.

**Dynamic Grid Approximation**   A standard method of finding approximate solutions to POMDPs is to discretize the belief space by covering it with a uniformly-spaced grid (otherwise called regular grid as shown in Figure 1, then solve an MDP that takes those grid points as states [1]. Unfortunately, the number of grid points required rises exponentially in the number of dimensions in the belief space, which corresponds to the number of states in the original space.

Recent studies have shown that in many cases, an agent actually travels through a very small subpart of its entire belief space. Roy and Gordon [9] find a low-dimensional subspace of the original belief space, then discretize that uniformly to get an MDP approximation to the original POMDP. This is an effective strategy, but it might be that the final uniform discretization is unnecessarily fine.

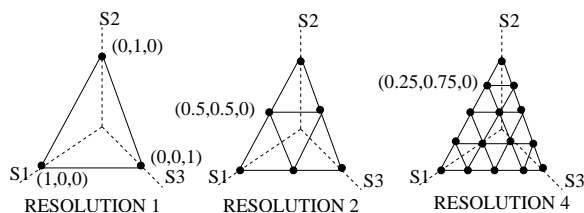

Figure 1: The figure depicts various regular dicretizations of a 3 dimensional belief simplex. The belief-space is the surface of the triangle, while grid points are the intersection of the lines drawn within the triangles. Using resolution of powers of 2 allows finer discretizations to include the points of coarser dicretizations.

In our work, we allocate grid points from a uniformly-spaced grid dynamically by simulating trajectories of the agent through the belief space. At each belief state experienced, we find the grid point that is closest to that belief state and add it to the set of grid points that we explicitly consider. In this way, we develop a set of grid points that is typically a very small subset of the entire possible grid, which is adapted to the parts of the belief space typically inhabited by the agent.

In particular, given a grid resolution $r$ and a belief state $b$ we can compute the coordinates (grid points $g_i$) of the belief simplex that contains $b$ using an efficient method called *Freudenthal* triangulation [2]. In addition to the vertices of a sub-simplex, Freundenthal triangulation also produces *barycentric* coordinates $\lambda_i$, with respect to $g_i$, which enable effective interpolation for the value of the belief state $b$ from the values of the grid points $g_i$ [1]. Using the barycentric coordinates we can also decide which is the closest grid-point to be added in the state space.

**Macro Actions**   The semi-Markov decision process (SMDP) model has become the preferred method for modeling temporally extended actions. An SMDP is defined as a five-tuple $(S,A,P,R,F)$, where $S$ is a finite set of states, $A$ is the set of actions, $P$ is the state

and action transition probability function, $R$ is the reward function, and $F$ is a function giving probability of transition times for each state-action pair. The transitions are at decision epochs only. The SMDP represents snapshots of the system at decision points, whereas the so-called *natural process* [8] describes the evolution of the system over all times. Discrete-time SMDPs represent transition distributions as $F(s', N|s, a)$, which specifies the expected number of steps $N$ that action $a$ will take before terminating in state $s'$ starting in state $s$. Q-learning generalizes nicely to discrete SMDPs. The Q-learning rule for discrete-time discounted SMDPs is

$$Q_{t+1}(s, a) \leftarrow (1 - \beta)Q_t(s, a) + \beta \left( R + \gamma^k \max_{a' \in A(s')} Q_t(s', a') \right),$$

where $\beta \in (0, 1)$, and action $a$ was initiated in state $s$, lasted for $k$ steps, and terminated in state $s'$, while generating a total discounted sum of rewards of $R$. Several frameworks for hierarchical reinforcement learning have been proposed, all of which are variants of SMDPs, such as the "options" framework [11].

Macro actions have been shown to be useful in a variety of MDP situations, but they have a special utility in POMDPs. For example, in a robot navigation task modeled as a POMDP, macro actions can consist of small state machines, such as a simple policy for driving down a corridor without hitting the walls until the end is reached. Such actions may have the useful property of reducing the entropy of the belief space, by helping a robot to localize its position. In addition, they relieve us of the burden of having to choose another primitive action based on the new belief state. Using macro actions tends to reduce the number of belief states that are visited by the agent. If a robot navigates largely by using macro-actions to move to important landmarks, it will never be necessary to model the belief states that are concerned with where the robot is within a corridor, for example.

**Algorithm** Our algorithm works by building a grid-based approximation of the belief space while executing a policy made up of macro actions. The policy is determined by "solving" the finite MDP over the grid points. Computing a policy over grid points equally spaced in the belief simplex, otherwise called regular discretization, is computationally intractable since the number of grid-points grows exponentially with the resolution [2]. Nonetheless, the value of a belief point in a regular dicretization can be interpolated efficiently from the values of the neighboring grid-points [2]. On the other hand, in variable resolution non-regular grids, interpolation can be computationally expensive [1]. A better approach is variable resolution with regular dicretization which takes advantage of fast interpolation and increases resolution only in the necessary areas [12]. Our approach falls in this last category with the addition of macro-actions, which exhibit various advantages over approaches using primitive actions only. Specifically, we use a reinforcement-learning algorithm (rather than dynamic programming) to compute a value function over the MDP states. It works by generating trajectories through the belief space according to the current policy, with some added exploration. Reinforcement learning using a model, otherwise called real time dynamic programming (RTDP) is not only better suited for huge spaces but in our case is also convenient in estimating the necessary models of our macro-actions over the experienced grid points.

While Figure 2 gives a graphical explanation of the algorithm, below we sketch the entire algorithm in detail:

1. Assume a current true state $s$. This is the physical true location of the agent, and it should have support in the current belief state $b$ (that is $b(s) \neq 0$).

2. Discretize the current belief state $b \rightarrow g_i$, where $g_i$ is the closest grid-point (with the maximum barycentric coordinate) in a regular discretization of the belief space. If $g_i$ is missing add it to the table. If the resolution is 1 initialize its value to zero otherwise interpolate its initial value from coarser resolutions.

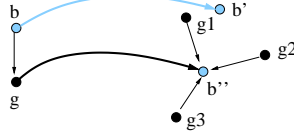

Figure 2: The agent finds itself at a belief state $b$. It maps $b$ to the grid point $g$, which has the largest barycentric coordinate among the sub-simplex coordinates that contain $b$. Now, it needs to do a value backup for that grid point. It chooses a macro action and executes it starting from the chosen grid-point, using the primitive actions and observations that it does along the way to update its belief state. It needs to get a value estimate for the resulting belief state $b''$. It does so by using the barycentric coordinates from the grid to interpolate a value from nearby grid points $g1$, $g2$, and $g3$. In case the nearest grid-point $g_i$ is missing, it is interpolated from coarser resolutions and added to the representation. If the resolution is 1, the value of $g_i$ is initialized to zero. The agent executes the macro-action from the same grid point $g$ multiple times so that it can approximate the probability distribution over the resulting belief-states $b''$. Finally, it can update the estimated value of the grid point $g$ and execute the macro-action chosen from the true belief state $b$. The process repeats from the next true belief state $b'$.

3. Choose a random action $\epsilon\%$ of the time. The rest of the time choose the best macro-action $\mu$ by interpolating over the Q values of the vertices of the sub-simplex that contains $b$: $\mu = \text{argmax}_{\mu \in \mathcal{M}} \sum_{i=1}^{|S|+1} \lambda_i Q(g_i, \mu)$.

4. Estimate $E\left[R(g_i, \mu) + \gamma^t V(b')\right]$ by sampling:

   (a) Sample a state $s$ from the current grid-belief state $g_i$ (which like all belief states represents a probability distribution over world states).

      i. Set $t = 0$
      ii. Choose the appropriate primitive action $a$ according to macro-action $\mu$.
      iii. Sample the next state $s'$ from the transition model $T(s, a, \cdot)$.
      iv. Sample an observation $z$ from observation model $O(a, s', \cdot)$.
      v. Store the reward $R(g_i, \mu) := R(g_i, \mu) + \gamma^t * R(s, a)$. For faster learning we use reward-shaping: $R(g_i, \mu) := R(g_i, \mu) + \gamma^{t+1} V(s') - \gamma^t V(s)$, where $V(s)$ are the values of the underlying MDP [5].
      vi. Update the belief state: $b'(j) := \frac{1}{\alpha} O(a, j, z) \sum_{i \in S} T(i, a, j)$, for all states $j$, where $\alpha$ is a normalizing factor.
      vii. Set $t = t+1, b = b', s = s'$ and repeat from step 4(a)ii until $\mu$ terminates.

   (b) Compute the value of the resulting belief state $b'$ by interpolating over the vertices in the resulting belief sub-simplex: $V(b') = \sum_{i}^{|S|+1} \lambda_i V(g_i)$. If the closest grid-point (with the maximum barycentric coordinate) is missing, interpolate it from coarser resolutions, and add it to the hash-table.

   (c) Repeat steps 4a and 4b multiple times, and average the estimate $\left[R(g_i, \mu) + \gamma^t V(b')\right]$.

5. Update the state action value: $Q(g_i, \mu) = (1 - \beta)Q(g_i, \mu) + \beta\left[R + \gamma^t V(b')\right]$.

6. Update the state value: $V(g_i) = \text{argmax}_{\mu \in \mathcal{M}} Q(g_i, \mu)$.

7. Execute the macro-action $\mu$ starting from belief state $b$ until termination. During execution, generate observations by sampling the POMDP model, starting from the true state $s$. Set $b = b'$ and $s = s'$ and go to step 2.

8. Repeat this learning epoch multiple times starting from the same $b$.

## 3 Experimental Results

We tested this algorithm by applying it to the problem of robot navigation, which is a classic sequential decision-making problem under uncertainty. We performed experiments in a corridor environment, shown in Figure 3. Such a topological map can be compiled into POMDPs, in which the discrete states stand for regions in the robot's pose space (for example 2 square meters in position and 90 degrees in orientation). In such a representation, the robot can move through the different environment states by taking actions such as "go-forward", "turn-left", and "turn-right". A macro-actions is implemented as a behavior (could be a POMDP policy) that takes as inputs observations and outputs actions. In our experiments we have a macro-action for going down the corridor until the end. In this navigation domain, our robot can only perceive sixteen possible observations, which indicate the presence of a wall and opening on the four sides of the robot. The observations are extracted from trained neural nets where the inputs are local occupancy grids constructed from sonar sensors and outputs are probabilities of walls and openings [4]. The POMDP model of the corridor environment has a reward function with value -1 in every state, except for -100 for going forward into a wall and +100 for taking any action from the four-way junction.

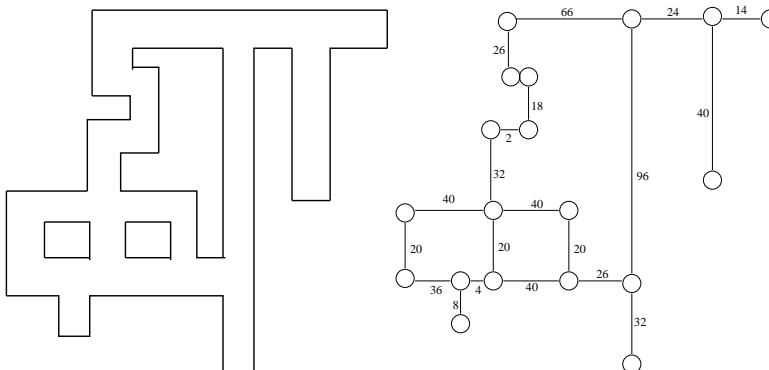

Figure 3: The figure on the left shows the floor plan of our experimental environment. The figure on the right is a topological map representation of the floor, which compiles into a POMDP with 1068 world states. The numbers next to the edges are the distances between the nodes in meters.

We ran the algorithm starting with resolution 1. When the average number of training steps stabilized we increased the resolution by multiplying it by 2. The maximum resolution we considered was 4. Each training episode started from the uniform initial belief state and was terminated when the four-way junction was reached or when more than 200 steps were taken. We ran the algorithm with and without the macro-action go-down-the-corridor. We compared the results with the *QMDP* heuristic which first solves the underlying MDP and then given any belief state, chooses the action that maximizes the dot product of the belief and Q values of state action pairs: $QMDP_a = \text{argmax}_a \sum_{s=1}^{|S|} b(s)Q(s,a)$.

**With Reward Shaping** The learning results in Figure 4 demonstrate that learning with macro-actions requires fewer number of training steps, which means the agent is getting to the goal faster. An exception is when the resolution is 1, where training with only primitive actions requires a small number of steps too. Nonetheless as we increase the resolution, training with primitive actions only does not scale well, because the number of states increases dramatically.

In general, the number of grid points used with or without macro-actions is significantly

smaller than the total number of points allowed for regular dicretization. For example, for a regular discretization the number of grid points can be computed by the formula given in [2], $\frac{(r+|S|-1)!}{r!(|S|-1)!}$, which is $5.4^{10}$ for $r = 4$ and $|S| = 1068$. Our algorithm with macro actions uses only about about 3000 and with primitive actions only about 6500 grid points.

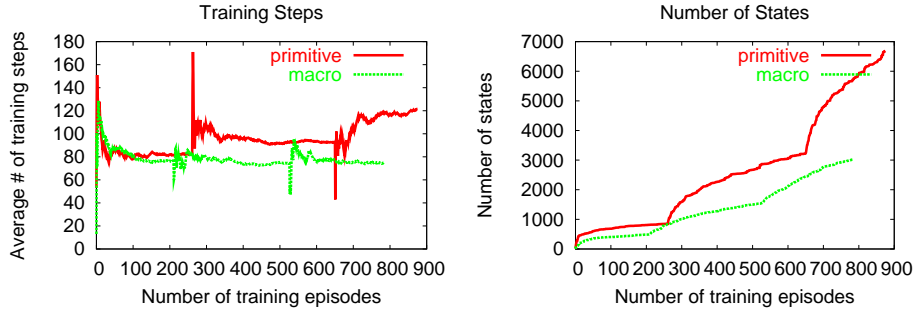

Figure 4: The graph on the left shows the average number of training-steps per episode as a function of the number of episodes. The graph on the right shows the number of grid-points added during learning. The sharp changes in the graph are due to the resolution increase.

We tested the policies that resulted from each algorithm by starting from a uniform initial belief state and a uniformly randomly chosen world state and simulating the greedy policy derived by interpolating the grid value function. We tested our plans over 200 different sampling sequences and report the results in Figure 5. A run was considered a success if the robot was able to reach the goal in fewer than 200 steps.

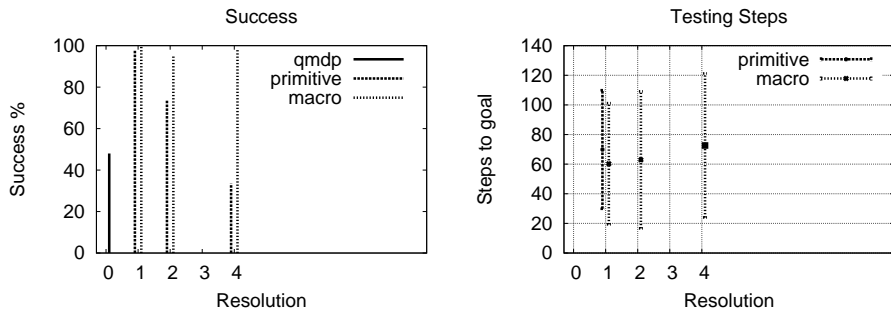

Figure 5: The figure on the left shows the success percentage for the different methods during testing. The results are reported after training for each resolution. The graph on the right shows the number of steps during testing. For the primitive-actions only algorithm we report the result for resolution 1 only, since it was as successful as the macro-action algorithm.

From Figure 5 we can conclude that the QMDP approach can never be $100\%$ successful, while the primitive-actions algorithm can perform quite well with resolution 1 in this environment. It is also evident from Figure 5 that as we increase the resolution, the macro-action algorithm maintains its robustness while the primitive-action algorithm performs considerably worse, mainly due to the fact that it requires more grid-points. In addition, when we compared the average number of testing steps for resolution 1 the macro-action algorithm seems to have learned a better policy. The macro-action policy policy seems to get worse for resolution 4 due to the increasing number of grid-points added in the repre-

sentation. This means that more training is required.

**Without Reward Shaping**   We also performed experiments to investigate the effect of reward-shaping. Figure 6 shows that with primitive actions only, the algorithm fails completely. However, with macro-actions the algorithm still converges and is more successful than the QMDP-heuristic.

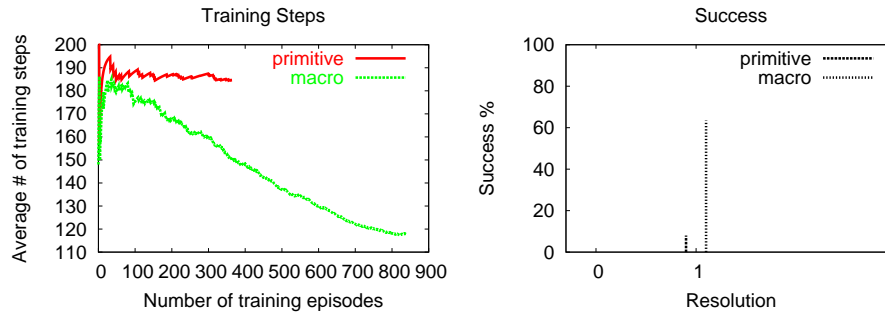

Figure 6: The The graph on the left shows the average number of training-steps (without reward shaping). The figure on the right shows the success percentage

**Information Gathering**   Apart from simulated experiments we also wanted to compare the performance of QMDP with the macro-action algorithm on a platform more closely related to a real robot. We used the Nomad 200 simulator and describe a test in Figure 7 to demonstrate how our algorithm is able to perform information gathering, as compared to QMDP.

## 4   Conclusions

In this paper we have presented an approximate planning algorithm for POMDPs that uses macro-actions. Our algorithm is able to solve a difficult planning problem, namely the task of navigating to a goal in a huge space POMDP starting from a **uniform** initial belief, which is more difficult than many of the tasks that similar algorithms are tested on. In addition, we have presented an effective reward-shaping approach to POMDPs that results in faster training (even without macro-actions).

In general macro-actions in POMDPs allow us to experience a smaller part of the state space, backup values faster, and do information gathering. As a result we can afford to allow for higher grid resolution which results in better performance. We cannot do this with only primitive actions (unless we use reward shaping) and it is completely out of the question for exact solution over the entire regular grid. In our current research we are investigating methods for dynamic discovery of "good" macro-actions given a POMDP.

## References

[1] M. Hauskrecht. Value-function approximations for partially observable Markov decision processes. *Journal of Artificial Intelligence Research*, 13:33–94, 2000.

[2] W. S. Lovejoy. Computationally feasible bounds for partially observed Markov decision processes. *Operations Research*, 39(1):162–175, January-February 1991.

[3] O. Madani, S. Hanks, and A. Gordon. On the undecidability of probabilistic planning and infinite-horizon partially observable Markov decision processes. In *Proceedings of the Sixteenth National Conference in Artificial Intelligence*, pages 409–416, 1999.

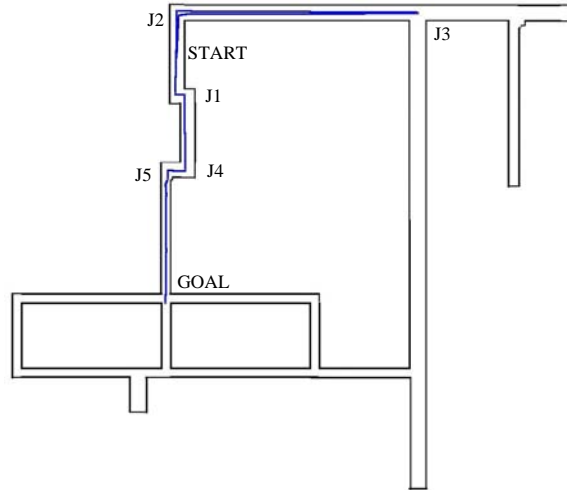

Figure 7: The figure shows the actual floor as it was designed in the Nomad 200 simulator. For the QMDP approach the robot starts from START with uniform initial belief. After reaching J2 the belief becomes bi-modal concentrating on J1 and J2. The robot then keeps turning left and right. On the other hand, with our planning algorithm, the robot again starts from START and a uniform initial belief. Upon reaching J2 the belief becomes bimodal over J1 and J2. The agent resolves its uncertainty by deciding that the best action to take is the go-down-the-corridor macro, at which point it encounters J3 and localizes. The robot then is able to reach its goal by traveling from J3, to J2 , J1, J4, and J5.

[4]  S. Mahadevan, G. Theocharous, and N. Khaleeli.  Fast concept learning for mobile robots. *Machine Learning and Autonomous Robots Journal (joint issue)*, 31/5:239–251, 1998.

[5]  A. Y. Ng, D. Harada, and S. Russell. Theory and application to reward shaping. In *Proceedings of the Sixteenth International Conference on Machine Learning*, 1999.

[6]  C. Papadimitriou and J. Tsitsiklis. The complexity of Markov decision processes. *Mathematics of Operation Research*, 12(3), 1987.

[7]  J. Pineau, G. Gordon, and S. Thrun.  Point-based value iteration: An anytime algorithm for POMDPs. In *International Joint Conference on Artificial Intelligence*, 2003.

[8]  M. Puterman. *Markov Decision Processes: Discrete Dynamic Stochastic Programming*. John Wiley, 1994.

[9]  N. Roy and G. Gordon.  Exponential family PCA for belief compression in POMDPs.  In *Advances in Neural Information Processing Systems*, 2003.

[10]  S. J. Russell and P. Norvig. *Artificial Intelligence: A Modern Approach*. Prentice Hall, 2nd edition, 2003.

[11]  R. S. Sutton, D. Precup, and S. Singh. Between MDPs and semi-MDPs: A framework for temporal abstraction in reinforcement learning. *Artificial Intelligence*, pages 112:181–211, 1999.

[12]  R. Zhou and E. A. Hansen. An improved grid-based approximation algorithm for POMDPs. In *Proceedings of the Seventeenth International Conference in Artificial intelligence (IJCAI-01)*, Seattle, WA, August 2001.
